# Developing Population Codes By Minimizing Description Length

**Richard S. Zemel**
CNL, The Salk Institute
10010 North Torrey Pines Rd.
La Jolla, CA   92037

**Geoffrey E. Hinton**
Department of Computer Science
University of Toronto
Toronto   M5S 1A4   Canada

## Abstract

The Minimum Description Length principle (MDL) can be used to train the hidden units of a neural network to extract a representation that is cheap to describe but nonetheless allows the input to be reconstructed accurately. We show how MDL can be used to develop highly redundant population codes. Each hidden unit has a location in a low-dimensional *implicit* space. If the hidden unit activities form a bump of a standard shape in this space, they can be cheaply encoded by the center of this bump. So the weights from the input units to the hidden units in an autoencoder are trained to make the activities form a standard bump. The coordinates of the hidden units in the implicit space are also learned, thus allowing flexibility, as the network develops a discontinuous topography when presented with different input classes. Population-coding in a space other than the input enables a network to extract nonlinear higher-order properties of the inputs.

Most existing unsupervised learning algorithms can be understood using the Minimum Description Length (MDL) principle (Rissanen, 1989). Given an ensemble of input vectors, the aim of the learning algorithm is to find a method of coding each input vector that minimizes the total cost, in bits, of communicating the input vectors to a receiver. There are three terms in the total description length:

- The **code-cost** is the number of bits required to communicate the code that the algorithm assigns to each input vector.

- The **model-cost** is the number of bits required to specify how to reconstruct input vectors from codes (e.g., the hidden-to-output weights).

- The **reconstruction-error** is the number of bits required to fix up any errors that occur when the input vector is reconstructed from its code.

Formulating the problem in terms of a communication model allows us to derive an objective function for a network (note that we are not actually sending the bits). For example, in competitive learning (vector quantization), the code is the identity of the winning hidden unit, so by limiting the system to $\mathcal{H}$ units we limit the average code-cost to at most $\log_2 \mathcal{H}$ bits. The reconstruction-error is proportional to the squared difference between the input vector and the weight-vector of the winner, and this is what competitive learning algorithms minimize. The model-cost is usually ignored.

The representations produced by vector quantization contain very little information about the input (at most $\log_2 \mathcal{H}$ bits). To get richer representations we must allow many hidden units to be active at once and to have varying activity levels. Principal components analysis (PCA) achieves this for linear mappings from inputs to codes. It can be viewed as a version of MDL in which we limit the code-cost by only having a few hidden units, and ignoring the model-cost and the accuracy with which the hidden activities must be coded. An autoencoder (see Figure 2) that tries to reconstruct the input vector on its output units will perform a version of PCA if the output units are linear. We can obtain novel and interesting unsupervised learning algorithms using this MDL approach by considering various alternative methods of communicating the hidden activities. The algorithms can all be implemented by backpropagating the derivative of the code-cost for the hidden units in addition to the derivative of the reconstruction-error backpropagated from the output units.

Any method that communicates each hidden activity separately and independently will tend to lead to *factorial* codes because any mutual information between hidden units will cause redundancy in the communicated message, so the pressure to keep the message short will squeeze out the redundancy. In (Zemel, 1993) and (Hinton and Zemel, 1994), we present algorithms derived from this MDL approach aimed at developing factorial codes. Although factorial codes are interesting, they are not robust against hardware failure nor do they resemble the population codes found in some parts of the brain. Our aim in this paper is to show how the MDL approach can be used to develop population codes in which the activities of hidden units are highly correlated. For a more complete discussion of the details of this algorithm, see (Zemel, 1993).

Unsupervised algorithms contain an implicit assumption about the nature of the structure or constraints underlying the input set. For example, competitive learning algorithms are suited to datasets in which each input can be attributed to one of a set of possible causes. In the algorithm we present here, we assume that each input can be described as a point in a low-dimensional continuous *constraint space*. For instance, a complex shape may require a detailed representation, but a set of images of that shape from multiple viewpoints can be concisely represented by first describing the shape, and then encoding each instance as a point in the constraint space spanned by the viewing parameters. Our goal is to find and represent the constraint space underlying high-dimensional data samples.

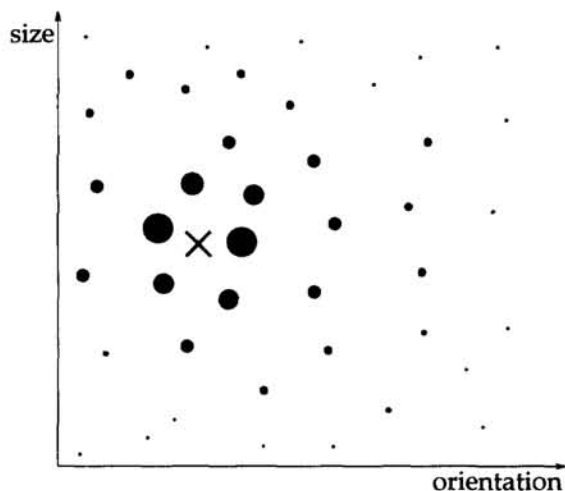

Figure 1: The population code for an instance in a two-dimensional implicit space. The position of each blob corresponds to the position of a unit within the population, and the blob size corresponds to the unit's activity. Here one dimension describes the size and the other the orientation of a shape. We can determine the instantiation parameters of this particular shape by computing the center of gravity of the blob activities, marked here by an "X".

# 1   POPULATION CODES

In order to represent inputs as points drawn from a constraint space, we choose a *population code* style of representation. In a population code, each code unit is associated with a position in what we call the *implicit space*, and the code units' pattern of activity conveys a single point in this space. This implicit space should correspond to the constraint space. For example, suppose that each code unit is assigned a position in a two-dimensional implicit space, where one dimension corresponds to the size of the shape and the second to its orientation in the image (see Figure 1). A population of code units broadly-tuned to different positions can represent any particular instance of the shape by their relative activity levels.

This example illustrates that population codes involve *three* quite different spaces: the input-vector space (the pixel intensities in the example); the hidden-vector space (where each hidden, or code unit entails an additional dimension); and this third, low-dimensional space which we term the implicit space. In a learning algorithm for population codes, this implicit space is intended to come to smoothly represent the underlying dimensions of variability in the inputs, i.e., the constraint space. For instance, the Kohonen (1982) algorithm defines the implicit space topology through fixed neighborhood relations, and the algorithm then manipulates hidden-vector space so that neighbors in implicit space respond to similar inputs.

This form of coding has several computational advantages, in addition to its significance due to its prevalence in biological systems. Population codes contain some redundancy and hence have some degree of fault-tolerance, and they reflect underlying structure of the input, in that similar inputs are mapped to nearby implicit positions. They also possess a hyperacuity property, as the number of implicit positions that can be represented far exceeds the number of code units.

## 2    LEARNING POPULATION CODES WITH MDL

Autoencoders are a general way of addressing issues of coding, in which the hidden unit activities for an input are the codes for that input which are produced by the input-hidden weights, and in which reconstruction from the code is done by the hidden-output mapping. In order to allow an autoencoder to develop population codes for an input set, we need some additional structure in the hidden layer that will allow a code vector to be interpreted as a point in implicit space. While most topographic-map formation algorithms (e.g., the Kohonen and elastic net (Durbin and Willshaw, 1987) algorithms) define the topology of this implicit space by fixed neighborhood relations, in our algorithm we use a more explicit representation. Each hidden unit has weights coming from the input units that determine its activity level. But in addition to these weights, it has another set of adjustable parameters that represent its coordinates in the implicit space. To determine what implicit position is represented by a vector of hidden activities, we can average together the implicit coordinates of the hidden units, weighting each coordinate vector by the activity level of the unit.

Suppose, for example, that each hidden unit is connected to an 8x8 retina and has 2 implicit coordinates that represent the size and orientation of a particular kind of shape on the retina, as in our earlier example. If we plot the hidden activity levels in the implicit space (not the input space), we would like to see a bump of activity of a standard shape (e.g., a Gaussian) whose center represents the instantiation parameters of the shape (Figure 2 depicts this for a 1D implicit space). If the activities form a perfect Gaussian bump of fixed variance we can communicate them by simply communicating the coordinates of the mean of the Gaussian; this is very economical if there are many less implicit coordinates than hidden units.

It is important to realize that the activity of a hidden unit is actually caused by the input-to-hidden weights, but by setting these weights appropriately we can make the activity match the height under the Gaussian in implicit space. If the activity bump is not quite perfect, we must also encode the *bump-error*—the misfit between the actual activity levels and the levels predicted by the Gaussian bump. The cost of encoding this misfit is what forces the activity bump in implicit space to approximate a Gaussian.

The reconstruction-error is then the deviation of the output from the input. This reconstruction ignores implicit space; the output activities only depend on the vector of hidden activities and weights.

### 2.1    The objective function

Currently, we ignore the model-cost, so the description length to be minimized is:

$$
\begin{aligned}
E^t &= B^t + R^t \\
&= \sum_{j=1}^{\mathcal{H}} (b_j^t - \hat{b}_j^t)^2 / 2V_B + \sum_{k=1}^{N} (a_k^t - c_k^t)^2 / 2V_R
\end{aligned}
\tag{1}
$$

where $a, b, c$ are the activities of units in the input, hidden, and output layers, respectively, $V_B$ and $V_R$ are the fixed variances of the Gaussians used for coding the

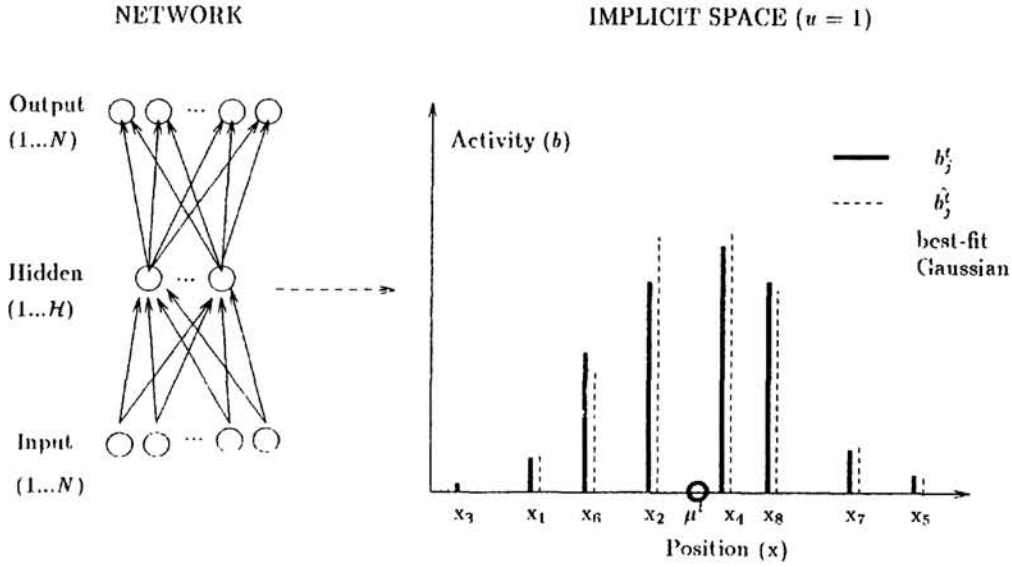

Figure 2: Each of the $\mathcal{H}$ hidden units in the autoencoder has an associated position in implicit space. Here we show a 1D implicit space. The activity $b_j^t$ of each hidden unit $j$ on case $t$ is shown by a solid line. The network fits the best Gaussian to this pattern of activity in implicit space. The predicted activity, $\hat{b}_j^t$, of unit $j$ under this Gaussian is based on the distance from $\mathbf{x_j}$ to the mean $\mu^t$; it serves as a target for $b_j^t$.

bump-errors and the reconstruction-errors, and the other symbols are explained in the caption of Figure 2.

We compute the actual activity of a hidden unit, $b_j^t$, as a normalized exponential of its total input.[1] Note that a unit's actual activity is independent of its position in implicit space. Its expected activity is its normalized value under the predicted Gaussian bump:

$$\hat{b}_j^t = \exp(-(\mathbf{x_j} - \mu^t)^2/2\sigma^2)/\sum_{i=1}^{\mathcal{H}}\exp(-(\mathbf{x_i} - \mu^t)^2/2\sigma^2) \tag{2}$$

where $\sigma$ is the width of the bump, which we assume for now is fixed throughout training.

We have explored several methods for computing the mean of this bump. Simply computing the center of gravity of the representation units' positions, weighted by their activity, produces a bias towards points in the center of implicit space. Instead, on each case, a separate minimization determines $\mu^t$; it is the position in implicit space that minimizes $B^t$ given $\{\mathbf{x_j}, b_j^t\}$. The network has full inter-layer connectivity, and linear output units. Both the network weights and the implicit coordinates of the hidden units are adapted to minimize $E$.

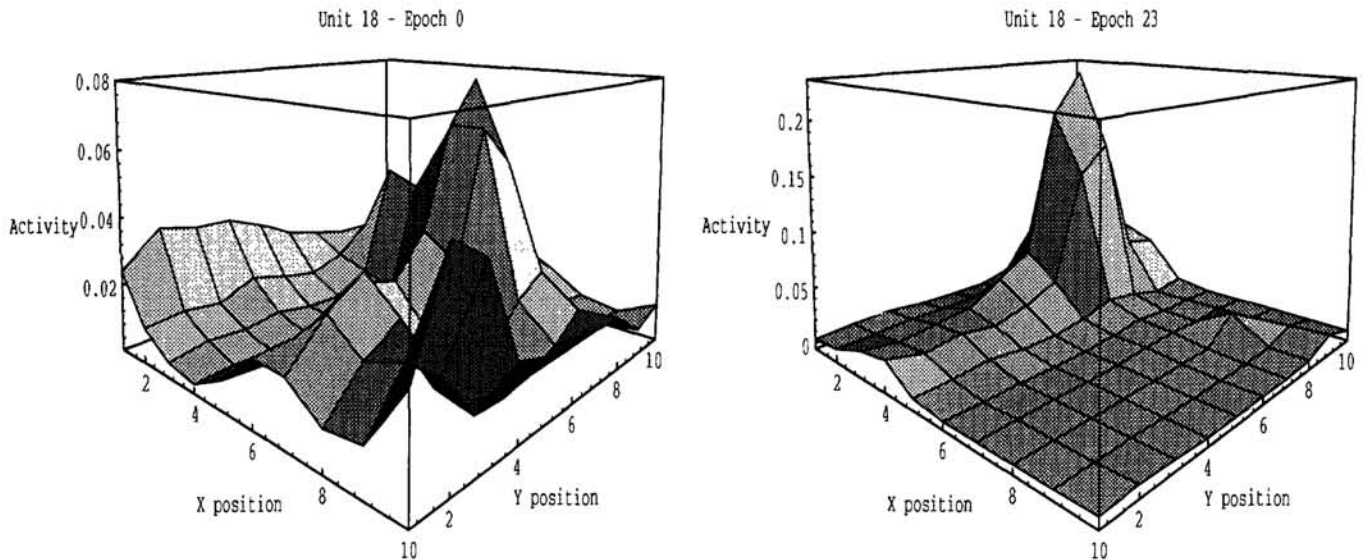

Figure 3: This figure shows the receptive field in implicit space for a hidden unit. The left panel shows that before learning, the unit responds randomly to 100 different test patterns, generated by positioning a shape in the image at each point in a 10x10 grid. Here the 2 dimensions in implicit space correspond to $x$ and $y$ positions. The right panel shows that after learning, the hidden unit responds to objects in a particular position, and its activity level falls off smoothly as the object position moves away from the center of the learned receptive field.

## 3    EXPERIMENTAL RESULTS

In the first experiment, each 8x8 real-valued input image contained an instance of a simple shape in a random $(x, y)$-position. The network began with random weights, and each of 100 hidden units in a random 2D implicit position; we trained it using conjugate gradient on 400 examples. The network converged after 25 epochs. Each hidden unit developed a receptive field so that it responded to inputs in a limited neighborhood that corresponded to its learned position in implicit space (see Figure 3). The set of hidden units covered the range of possible positions.

In a second experiment, we also varied the orientation of the shape and we gave each hidden unit three implicit coordinates. The network converged after 60 epochs of training on 1000 images. The hidden unit activities formed a population code that allowed the input to be accurately reconstructed.

A third experiment employed a training set where each image contained either a horizontal or vertical bar, in some random position. The hidden units formed an interesting 2D implicit space in this case: one set of hidden units moved to one corner of the space, and represented instances of one shape, while the other group moved to an opposite corner and represented the other (Figure 4). The network was thus able to squeeze a third dimension (i.e., which shape) into the 2D implicit space. This type of representation would be difficult to learn in a Kohonen network; the fact that the hidden units *learn* their implicit coordinates allows more flexibility than a system in which these coordinates are fixed in advance.

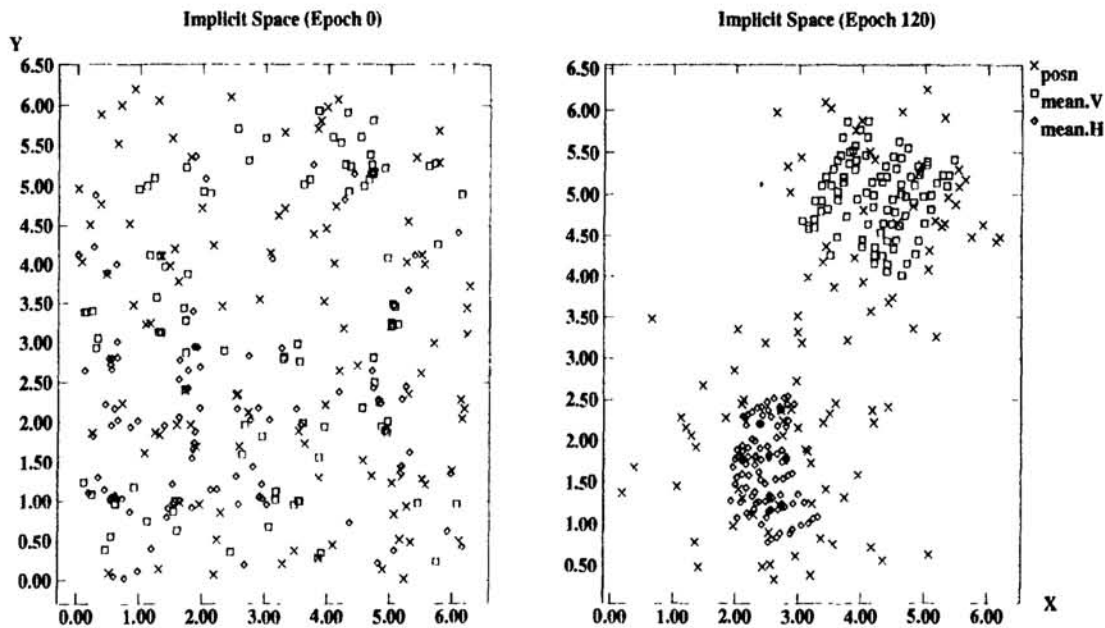

Figure 4: This figure shows the positions of the hidden units and the means in the 2D implicit space before and after training on the horizontal/vertical task. The means in the top right of the second plot all correspond to images containing vertical bars, while the other set correspond to horizontal bar images. Note that some hidden units are far from all the means; these units do not play a role in the coding of the input, and are free to be recruited for other types of input cases.

## 4    RELATED WORK

This new algorithm bears some similarities to several earlier algorithms. In the experiments presented above, each hidden unit learns to act as a Radial Basis Function (RBF) unit. Unlike standard RBFs, however, here the RBF activity serves as a target for the activity levels, and is determined by distance in a space other than the input space.

Our algorithm is more similar to topographic map formation algorithms, such as the Kohonen and elastic-net algorithms. In these methods, however, the population-code is in effect formed in input space. Population coding in a space other than the input enables our networks to extract nonlinear higher-order properties of the inputs.

In (Saund, 1989), hidden unit patterns of activity in an autoencoder are trained to form Gaussian bumps, where the center of the bump is intended to correspond to the position in an underlying dimension of the inputs. In addition to the objective functions being quite different in the two algorithms, another crucial difference exists: in his algorithm, as well as the other earlier algorithms, the implicit space topology is statically determined by the ordering of the hidden units, while units in our model learn their implicit coordinates.

## 5    CONCLUSIONS AND CURRENT DIRECTIONS

We have shown how MDL can be used to develop non-factorial, redundant representations. The objective function is derived from a communication model where rather than communicating each hidden unit activity independently, we instead communicate the location of a Gaussian bump in a low-dimensional implicit space. If hidden units are appropriately tuned in this space their activities can then be inferred from the bump location.

Our method can easily be applied to networks with multiple hidden layers, where the implicit space is constructed at the last hidden layer before the output and derivatives are then backpropagated; this allows the implicit space to correspond to arbitrarily high-order input properties. Alternatively, instead of using multiple hidden layers to extract a single code for the input, one could use a hierarchical system in which the code-cost is computed at every layer.

A limitation of this approach (as well as the aforementioned approaches) is the need to predefine the dimensionality of implicit space. We are currently working on an extension that will allow the learning algorithm to determine for itself the appropriate number of dimensions in implicit space. We start with many dimensions but include the cost of specifying $\mu^t$ in the description length. This obviously depends on how many implicit coordinates are used. If all of the hidden units have the same value for one of the implicit coordinates, it costs nothing to communicate that value for each bump. In general, the cost of an implicit coordinate depends on the ratio between its variance (over all the different bumps) and the accuracy with which it must be communicated. So the network can save bits by reducing the variance for unneeded coordinates. This creates a smooth search space for determining how many implicit coordinates are needed.

### Acknowledgements
This research was supported by grants from NSERC, the Ontario Information Technology Research Center, and the Institute for Robotics and Intelligent Systems. Geoffrey Hinton is the Noranda Fellow of the Canadian Institute for Advanced Research. We thank Peter Dayan for helpful discussions.

## Footnotes

[1] $b_j^t = \exp(net_j^t)/\sum_{i=1}^{H}\exp(net_i^t)$, where $net_j^t$ is the net input into unit $j$ on case $t$.

### References
Durbin, R. and Willshaw, D. (1987). An analogue approach to the travelling salesman problem. *Nature*, 326:689-691.

Hinton, G. and Zemel, R. (1994). Autoencoders, minimum description length, and Helmholtz free energy. To appear in Cowan, J.D., Tesauro, G., and Alspector, J. (eds.), *Advances in Neural Information Processing Systems 6*. San Francisco, CA: Morgan Kaufmann.

Kohonen, T. (1982). Self-organized formation of topologically correct feature maps. *Biological Cybernetics*, 43:59-69.

Rissanen, J. (1989). *Stochastic Complexity in Statistical Inquiry.* World Scientific Publishing Co., Singapore.

Saund, E. (1989). Dimensionality-reduction using connectionist networks. *IEEE Transactions on Pattern Analysis and Machine Intelligence*, 11(3):304-314.

Zemel, R. (1993). *A Minimum Description Length Framework for Unsupervised Learning.* Ph.D. Thesis, Department of Computer Science, University of Toronto.